# The Missing Link - A Probabilistic Model of Document Content and Hypertext Connectivity

**David Cohn**
Burning Glass Technologies
201 South Craig St, Suite 2W
Pittsburgh, PA 15213
*david.cohn@burning-glass.com*

**Thomas Hofmann**
Department of Computer Science
Brown University
Providence, RI 02192
*th@cs.brown.edu*

## Abstract

We describe a joint probabilistic model for modeling the contents and inter-connectivity of document collections such as sets of web pages or research paper archives. The model is based on a probabilistic factor decomposition and allows identifying principal topics of the collection as well as authoritative documents within those topics. Furthermore, the relationships between topics is mapped out in order to build a predictive model of link content. Among the many applications of this approach are information retrieval and search, topic identification, query disambiguation, focused web crawling, web authoring, and bibliometric analysis.

## 1 Introduction

No text, no paper, no book can be isolated from the all-embracing corpus of documents it is embedded in. Ideas, thoughts, and work described in a document inevitably relate to and build upon previously published material.[1] Traditionally, this interdependency has been represented by citations, which allow authors to explicitly make references to related documents. More recently, a vast number of documents have been "published" electronically on the world wide web; here, interdependencies between documents take the form of hyperlinks, and allow instant access to the referenced material. We would like to have some way of modeling these interdependencies, to understand the structure implicit in the contents and connections of a given document base without resorting to manual clustering, classification and ranking of documents.

The main goal of this paper is to present a joint probabilistic model of document content and connectivity, i.e., a parameterized stochastic process which mimics the generation of documents as part of a larger collection, and which could make accurate predictions about the existence of hyperlinks and citations. More precisely, we present an extension of our work on Probabilistic Latent Semantic Analysis (PLSA) [4, 7] and Probabilistic HITS (PHITS) [3, 8] and propose a mixture model to perform a simultaneous decomposition of the contingency tables associated with word occurrences and citations/links into "topic" factors. Such a model can be extremely useful in many applications, a few of which are:

- Identifying topics and common subjects covered by documents. Representing

documents in a low-dimensional space can help understanding of relations between documents and the topics they cover. Combining evidence from terms and links yields potentially more meaningful and stable factors and better predictions.

- Identifying authoritative documents on a given topic. The authority of a document is correlated with how frequently it is cited, and by whom. Identifying topic-specific authorities is a key problems for search engines [2].

- Predictive navigation. By predicting what content might be found "behind" a link, a content/connectivity model directly supports navigation in a document collection, either through interaction with human users or for intelligent spidering.

- Web authoring support. Predictions about links based on document contents can support authoring and maintenance of hypertext documents, e.g., by (semi-) automatically improving and updating link structures.

These applications address facets of one of the most pressing challenges of the "information age": how to locate useful information in a semi-structured environment like the world wide web. Much of this difficulty, which has led to the emergence of an entire new industry, is due to the impoverished explicit structure of the web as a whole. Manually created hyperlinks and citations are limited in scope – the annotator can only add links and pointers to other document they are aware of and have access to. Moreover, these links are static; once the annotator creates a link between documents, it is unchanging. If a different, more relevant document appears (or if the cited document disappears), the link may not get updated appropriately. These and other deficiencies make the web inherently "noisy" – links between relevant documents may not exist and existing links might sometimes be more or less arbitrary. Our model is a step towards a technology that will allow us to dynamically infer more reliable inter-document structure from the impoverished structure we observe.

In the following section, we first review PLSA and PHITS. In Section 3, we show how these two models can be combined into a joint probabilistic term-citation model. Section 4 describes some of the applications of this model, along with preliminary experiments in several areas. In Section 5 we consider future directions and related research.

## 2   PLSA and PHITS

PLSA [7] is a statistical variant of Latent Semantic Analysis (LSA) [4] that builds a factored multinomial model based on the assumption of an underlying document generation process. The starting point of (P)LSA is the term-document matrix $N$ of word counts, i.e., $N_{ij}$ denotes how often a term (single word or phrase) $t_i$ occurs in document $d_j$. In LSA, $N$ is decomposed by a SVD and factors are identified with the left/right principal eigenvectors. In contrast, PLSA performs a probabilistic decomposition which is closely related to the non-negative matrix decomposition presented in [9]. Each factor is identified with a state $z_k$ ($1 \leq k \leq K$) of a latent variable with associated relative frequency estimates $P(t_i|z_k)$ for each term in the corpus. A document $d_j$ is then represented as a convex combination of factors with mixing weights $P(z_k|d_j)$, i.e., the predictive probabilities for terms in a particular document are constrained to be of the functional form $P(t_i|d_j) = \sum_k P(t_i|z_k)P(z_k|d_j)$, with non-negative probabilities and two sets of normalization constraints $\sum_i P(t_i|z_k) = 1$ for all $k$ and $\sum_k P(z_k|d_j) = 1$ for all $j$.

Both the factors and the document-specific mixing weights are learned by maximizing the likelihood of the observed term frequencies. More formally, PLSA aims at maximizing $\mathcal{L} = \sum_{i,j} N_{ij} \log \sum_k P(t_i|z_k)P(z_k|d_j)$. Since factors $z_k$ can be interpreted as states of a latent mixing variable associated with each observation (i.e., word occurrence), the Expectation-Maximization algorithm can be applied to find a local maximum of $\mathcal{L}$.

PLSA has been demonstrated to be effective for *ad hoc* information retrieval, language

modeling and clustering. Empirically, different factors usually capture distinct "topics" of a document collection; by clustering documents according to their dominant factors, useful topic-specific document clusters often emerge (using the Gaussian factors of LSA, this approach is known as "spectral clustering").

It is important to distinguish the factored model used here from standard probabilistic mixture models. In a mixture model, each object (such as a document) is usually assumed to come from *one* of a set of latent sources (e.g. a document is either from $z_1$ or $z_2$). Credit for the object may be distributed among several sources because of ambiguity, but the model insists that only one of the candidate sources is the true origin of the object. In contrast, a factored model assumes that each object comes from a *mixture* of sources — without ambiguity, it can assert that a document is half $z_1$ and half $z_2$. This is because the latent variables are associated with each *observation* and not with each *document* (set of observations).

PHITS [3] performs a probabilistic factoring of document citations used for bibliometric analysis. Bibliometrics attempts to identify topics in a document collection, as well as influential authors and papers on those topics, based on patterns in citation frequency. This analysis has traditionally been applied to references in printed literature, but the same techniques have proven successful in analyzing hyperlink structure on the world wide web [8].

In traditional bibliometrics, one begins with a matrix $A$ of document-citation pairs. Entry $A_{ij}$ is nonzero if and only if document $d_i$ is cited by document $d_j$ or, equivalently, if $d_j$ contains a hyperlink to $d_i$.[2] The principal eigenvectors of $AA'$ are then extracted, with each eigenvector corresponding to a "community" of roughly similar citation patterns. The coefficient of a document in one of these eigenvectors is interpreted as the "authority" of that document within the community — how likely it is to by cited within that community. A document's coefficient in the principal eigenvectors of $A'A$ is interpreted as its "hub" value in the community — how many authoritative documents it cites within the community.

In PHITS, a probabilistic model replaces the eigenvector analysis, yielding a model that has clear statistical interpretations. PHITS is mathematically identical to PLSA, with one distinction: instead of modeling the citations contained within a document (corresponding to PLSA's modeling of terms in a document), PHITS models "inlinks," the citations *to* a document. It substitutes a citation-source probability estimate $P(c_l|z_k)$ for PLSA's term probability estimate. As with PLSA and spectral clustering, the principal factors of the model are interpreted as indicating the principal citation communities (and by inference, the principal topics). For a given factor/topic $z_k$, the probability that a document is cited, $P(d_j|z_k)$, is interpreted as the document's authority with respect to that topic.

## 3   A Joint Probabilistic Model for Content and Connectivity

Linked and hyperlinked documents are generally composed of terms *and* citations; as such, both term-based PLSA and citation-based PHITS analyses are applicable. Rather than applying each separately, it is reasonable to merge the two analyses into a joint probabilistic model, explaining terms and citations in terms of a common set of underlying factors.

Since both PLSA and PHITS are based on a similar decomposition, one can define the following joint model for predicting citations/links and terms in documents:

$$P(t_i|d_j) = \sum_k P(t_i|z_k)P(z_k|d_j), \quad P(c_l|d_j) = \sum_k P(c_l|z_k)P(z_k|d_j). \quad (1)$$

Notice that both decompositions share the same document-specific mixing proportions $P(z_k|d_j)$. This couples the conditional probabilities for terms and citations: each "topic"

has some probability $P(c_l|z_k)$ of linking to document $d_l$ as well as some probability $P(t_i|z_k)$ of containing an occurrence of term $t_i$. The advantage of this joint modeling approach is that it integrates content- and link–information in a principled manner. Since the mixing proportions are shared, the learned decomposition must be consistent with content *and* link statistics. In particular, this coupling allows the model to take evidence about link structure into account when making predictions about document content and vice versa. Once a decomposition is learned, the model may be used to address questions like "What words are likely to be found in a document with this link structure?" or "What link structure is likely to go with this document?" by simple probabilistic inference.

The relative importance one assigns to predicting terms and links will depend on the specific application. In general, we propose maximizing the following (normalized) log–likelihood function with a relative weight $\alpha$.

$$
\mathcal{L} = \sum_j \left[ \alpha \sum_i \frac{N_{ij}}{\sum_{i'} N_{i'j}} \log \sum_k P(t_i|z_k) P(z_k|d_j) \right.
$$
$$
\left. + (1-\alpha) \sum_l \frac{A_{lj}}{\sum_{l'} A_{l'j}} \log \sum_k P(c_l|z_k) P(z_k|d_j) \right] \quad (2)
$$

The normalization by term/citation counts ensures that each document is given the same weight in the decomposition, regardless of the number of observations associated with it.

Following the EM approach it is straightforward to derive a set of re-estimation equations. For the E-step one gets formulae for the posterior probabilities of the latent variables associated with each observation[3]

$$
P(z_k|t_i, d_j) = \frac{P(t_i|z_k) P(z_k|d_k)}{P(t_i|d_j)}, \quad P(z_k|c_l, d_j) = \frac{P(c_l|z_k) P(z_k|d_j)}{P(c_l|d_j)}. \quad (3)
$$

The class-conditional distributions are recomputed in the M-step according to

$$
P(t_i|z_k) = \sum_j \frac{N_{ij}}{\sum_{i'} N_{i'j}} P(z_k|t_i, d_j), \quad P(c_l|z_k) = \sum_j \frac{A_{lj}}{\sum_{l'} A_{l'j}} P(z_k|c_l, d_j), \quad (4)
$$

along with the mixing proportions

$$
P(z_k|d_j) \propto \alpha \sum_i \frac{N_{ij}}{\sum_{i'} N_{i'j}} P(z_k|t_i, d_j) + (1-\alpha) \sum_l \frac{A_{lj}}{\sum_{l'} A_{l'j}} P(z_k|c_l, d_j). \quad (5)
$$

## 4  Experiments

In the introduction, we described many potential applications of the the joint probabilistic model. Some, like classification, are simply extensions of the individual PHITS and PLSA models, relying on the increased power of the joint model to improve their performance. Others, such as intelligent web crawling, are unique to the joint model and require its simultaneous modelling of a document's contents and connections.

In this section, we first describe experiments verifying that the joint model does yield improved classification compared with the individual models. We then describe a quantity called "reference flow" which can be computed from the joint model, and demonstrate its use in guiding a web crawler to pages of interest.

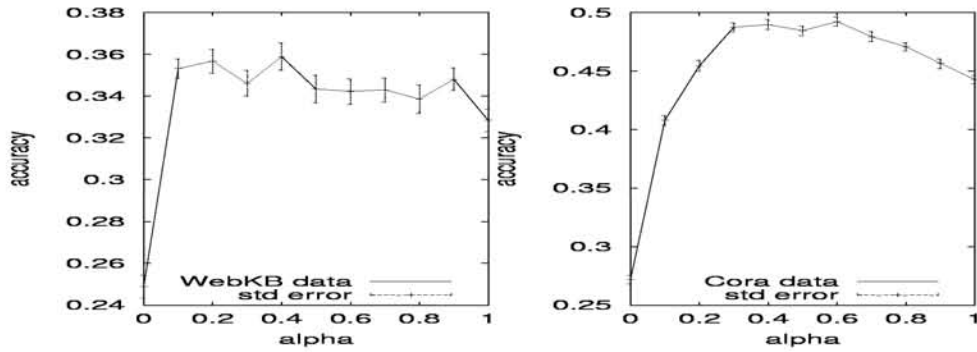

Figure 1: Classification accuracy on the WebKB and *Cora* data sets for PHITS ($\alpha = 0$), PLSA ($\alpha = 1$) and the joint model ($0 < \alpha < 1$).

We used two data sets in our experiments. The WebKB data set [11], consists of approximately 6000 web pages from computer science departments, classified by school and category (student, course, faculty, etc.). The *Cora* data set [10] consists of the abstracts and references of approximately 34,000 computer science research papers; of these, we used the approximately 2000 papers categorized into one of seven subfields of machine learning.

### 4.1 Classification

Although the joint probabilistic model performs unsupervised learning, there are a number of ways it may be used for classification. One way is to associate each document with its dominant factor, in a form of spectral clustering. Each factor is then given the label of the dominant class among its associated documents. Test documents are judged by whether their dominant factor shares their label.

Another approach to classification (but one that forgoes clustering) is a factored nearest neighbor approach. Test documents are judged against the label of their nearest neighbor, but the "nearest" neighbor is determined by cosines of their projections in factor space. This is the method we used for our experiments.

For the *Cora* and WebKB data, we used seven factors and six factors respectively, arbitrarily selecting the number to correspond to the number of human-derived classes. We compared the power of the joint model with that of the individual models by varying $\alpha$ from zero to one, with the lower and upper extremes corresponding to PHITS and PLSA, respectively. For each value of $\alpha$, a randomly selected 15% of the data were reserved as a test set. The models were tempered (as per [7]) with a lower limit of $\beta = 0.8$, decreasing $\beta$ by a factor of 0.9 each time the data likelihood stopped increasing.

Figure 1 illustrates several results. First, the accuracy of the joint model (where $\alpha$ is neither 0 nor 1), is greater than that of either model in isolation, indicating that the contents and link structure of a document collection do indeed corroborate each other. Second, the increase in accuracy is robust across a wide range of mixing proportions.

### 4.2 Reference Flow

The previous subsection demonstrated how the joint model amplifies abilities found in the individual models. But the joint model also provides features found in neither of its progenitors.

A document $d$ may be thought of as occupying a point $\vec{z} = \{P(z_1|d), \ldots, P(z_k|d)\}$ in the joint model's space of factor mixtures. The terms in $d$ act as "signposts" describing $\vec{z}$, and the links act as directed connections between that point and others. Together, they provide a *reference flow*, indicating a referential connection between one topic and another. This reference flow exists between arbitrary points in the factor space, even in the absence of documents that map directly to those points.

Consider a reference from document $d_i$ to document $d_j$, and two points in factor space $\vec{z}_m$ and $\vec{z}_n$, not particularly associated with $d_i$ or $d_j$. Our model allows us to compute $P(d_i|\vec{z}_m)$ and $P(d_j|\vec{z}_n)$, the probability that the combination of factors at $\vec{z}_m$ and $\vec{z}_n$ are responsible for $d_i$ and $d_j$ respectively. Their product $P(d_i|\vec{z}_m)P(d_j|\vec{z}_n)$ is then the prob-

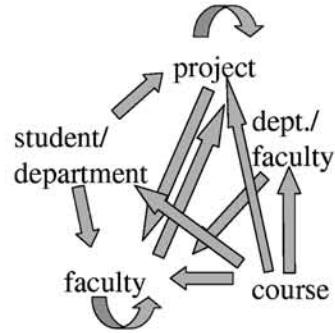

Figure 2: Principal reference flow between the primary topics identified in the examined subset of the *WebKB* archive.

ability that the observed link represents a reference between those two points in factor space. By integrating over all links in the corpus we can compute, $f_{mn} = \sum_{i,j:A_{ij}\neq 0} P(d_i|\vec{z}_m)P(d_j|\vec{z}_n)$, an unnormalized "reference flow" between $\vec{z}_m$ and $\vec{z}_n$. Figure 2 shows the principal reference flow between several topics in the WebKB archive.

### 4.3 Intelligent Web Crawling with Reference Flow

Let us suppose that we want to find new web pages on a certain topic, described by a set of words composed into a target pseudodocument $d_t$. We can project $d_t$ into our model to identify the point $\vec{z}_t$ in factor space that represents that topic. Now, when we explore web pages, we want to follow links that will lead us to new documents that also project to $\vec{z}_t$.

To do so, we can use reference flow. Consider a web page $d_s$ (or section of a web page[4]). Although we don't know where its links point, we do know what words it contains. We can project them as a peudodocument to find $\vec{z}_s$ the point in factor space the page/section occupies, prior to any information about its links. We can then use our model to compute the reference flow $f_{st}$ indicating the (un-normalized) probability that a document at $\vec{z}_s$ would contain a link to one at $\vec{z}_t$.

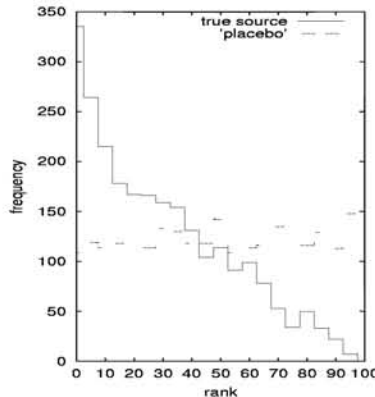

As a greedy solution, we could simply follow links in documents or sections that have the highest reference flow toward the target topic. Or if computation is no barrier, we could (in theory) use reference flow as state transition probabilities and find an optimal link to follow by

Figure 3: When ranked according to magnitude of reference flow to a designated target, a "true source" scores much higher than a placebo source document drawn at random.

treating the system as a continuous-state Markov decision process.

To test our model's utility in intelligent web crawling, we conducted experiments on the WebKB data set using the greedy solution. On each trial, a "target page" $d_t$ was selected at random from the corpus. One "source page" $d_s$ containing a link to the target was identified, and the reference flow $f_{st}$ computed. The larger the reference flow, the stronger our model's expectation that there is a directed link from the source to the target.

We ranked this flow against the reference flow to the target from 100 randomly chosen "distractor" pages $d_{r1}, d_{r2} \ldots, d_{r100}$. As seen in Figure 3, reference flow provides significant predictive power. Based on 2400 runs, the median rank for the "true source" was 27/100, versus a median rank of 50/100 for a "placebo" distractor chosen at random. Note that the distractors were not screened to ensure that they did not also contain links to the target; as such, some of the high-ranking distractors may also have been valid sources for the target in question.

## 5   Discussion and Related Work

There have been many attempts to combine link and term information on web pages, though most approaches are *ad hoc* and have been aimed at increasing the retrieval of authoritative documents relevant to a given query. Bharat and Henzinger [1] provide a good overview of research in that area, as well as an algorithm that computes bibliometric authority after weighting links based on the relevance of the neighboring terms. The machine learning community has also recently taken an interest in the sort of relational models studied by Bibliometrics. Getoor et al. [5] describe a general framework for learning probabilistic relational models from a database, and present experiments in a variety of domains.

In this paper, we have described a specific probabilistic model which attempts to explain both the contents and connections of documents in an unstructured document base. While we have demonstrated preliminary results in several application areas, this paper only scratches the surface of potential applications of a joint probabilistic document model.

## Footnotes

[1] Although the weakness of our memory might make us forget this at times.

[2]In fact, since multiple citations/links may exist, we treat $A_{ij}$ as a count variable.

[3]Our experiments used a tempered version of Equation 3 to minimize overfitting; see [7] for details.

[4]Though not described here, we have had success using our model for document segmentation, following an approach similar to that of [6]. By projecting successive n-sentence windows of a document into the factored model, we can observe its trajectory through "topic space." A large jump in the factor mixture between successive windows indicates a probable topic boundary in document.

## References

[1] K. Bharat and M. R. Henzinger. Improved algorithms for topic distillation in hyperlinked environments. In *Proceedings of the 21st Annual International ACM SIGIR Conference on Research and Development in Information Retrieval*, 1998.

[2] S. Brin and L. Page. The anatomy of a large-scale hypertextual web search engine. Technical report, Computer Science Department, Stanford University, 1998.

[3] D. Cohn and H. Chang. Learning to probabilistically identify authoritative documents. In *Proceedings of the 17th International Conference on Machine Learning*, 2000.

[4] S. Deerwester, S. T. Dumais, G. W. Furnas, T. K. Landauer, and R. Harshman. Indexing by latent semantic analysis. *J. of the American Society for Information Science*, 41:391–407, 1990.

[5] L. Getoor, N. Friedman, D. Koller, and A. Pfeffer. Learning probabilistic relational models. In S. Dzeroski and N. Lavrac, editors, *Relational Data Mining*. Springer-Verlag, 2001.

[6] M. Hearst. Multi-paragraph segmentation of expository text. In *Proceedings of ACL*, June 1994.

[7] T. Hofmann. Probabilistic latent semantic analysis. In *Proceedings of the 15th Conference on Uncertainty in AI*, pages 289–296, 1999.

[8] J. Kleinberg. Authoritative sources in a hyperlinked environment. In *Proc. 9th ACM-SIAM Symposium on Discrete Algorithms*, 1998.

[9] D. D. Lee and H. S. Seung. Learning the parts of objects by non-negative matrix factorization. *Nature*, pages 788–791, 1999.

[10] A. McCallum, K. Nigam, J. Rennie, and K. Seymore. Automating the construction of internet portals with machine learning. *Information Retrieval Journal*, 3:127–163, 2000.

[11] Web→KB. Available electronically at http://www.cs.cmu.edu/~WebKB/.
